# Unconstrained Online Handwriting Recognition with Recurrent Neural Networks

**Alex Graves**
TUM, Germany
alex@idsia.ch

**Santiago Fernández**
IDSIA, Switzerland
santiago@idsia.ch

**Marcus Liwicki**
University of Bern, Switzerland
liwicki@iam.unibe.ch

**Horst Bunke**
University of Bern, Switzerland
bunke@iam.unibe.ch

**Jürgen Schmidhuber**
IDSIA, Switzerland and TUM, Germany
juergen@idsia.ch

## Abstract

In online handwriting recognition the trajectory of the pen is recorded during writing. Although the trajectory provides a compact and complete representation of the written output, it is hard to transcribe directly, because each letter is spread over many pen locations. Most recognition systems therefore employ sophisticated preprocessing techniques to put the inputs into a more localised form. However these techniques require considerable human effort, and are specific to particular languages and alphabets. This paper describes a system capable of directly transcribing raw online handwriting data. The system consists of an advanced recurrent neural network with an output layer designed for sequence labelling, combined with a probabilistic language model. In experiments on an unconstrained online database, we record excellent results using either raw or preprocessed data, well outperforming a state-of-the-art HMM based system in both cases.

## 1  Introduction

Handwriting recognition is traditionally divided into offline and online recognition. Offline recognition is performed on images of handwritten text. In online handwriting the location of the pen-tip on a surface is recorded at regular intervals, and the task is to map from the sequence of pen positions to the sequence of words.

At first sight, it would seem straightforward to label raw online inputs directly. However, the fact that each letter or word is distributed over many pen positions poses a problem for conventional sequence labelling algorithms, which have difficulty processing data with long-range interdependencies. The problem is especially acute for unconstrained handwriting, where the writing style may be cursive, printed or a mix of the two, and the degree of interdependency is therefore difficult to determine in advance. The standard solution is to preprocess the data into a set of localised features. These features typically include geometric properties of the trajectory in the vicinity of every data point, pseudo-offline information from a generated image, and character level shape characteristics [6, 7]. Delayed strokes (such as the crossing of a 't' or the dot of an 'i') require special treatment because they split up the characters and therefore interfere with localisation. HMMs [6] and hybrid systems incorporating time-delay neural networks and HMMs [7] are commonly trained with such features.

The issue of classifying preprocessed versus raw data has broad relevance to machine learning, and merits further discussion. Using hand crafted features often yields superior results, and in some cases can render classification essentially trivial. However, there are three points to consider in favour of raw data. Firstly, designing an effective preprocessor requires considerable time and expertise. Secondly, hand coded features tend to be more task specific. For example, features designed

for English handwriting could not be applied to languages with substantially different alphabets, such as Arabic or Chinese. In contrast, a system trained directly on pen movements could be applied to any alphabet. Thirdly, using raw data allows feature extraction to be built into the classifier, and the whole system to be trained together. For example, convolutional neural networks [10], in which a globally trained hierarchy of network layers is used to extract progressively higher level features, have proved effective at classifying raw images, such as objects in cluttered scenes or isolated handwritten characters [15, 11]. (Note than convolution nets are less suitable for unconstrained handwriting, because they require the text images to be presegmented into characters [10]).

In this paper, we apply a recurrent neural network (RNN) to online handwriting recognition. The RNN architecture is bidirectional Long Short-Term Memory [3], chosen for its ability to process data with long time dependencies. The RNN uses the recently introduced connectionist temporal classification output layer [2], which was specifically designed for labelling unsegmented sequence data. An algorithm is introduced for applying grammatical constraints to the network outputs, thereby providing word level transcriptions. Experiments are carried out on the IAM online database [12] which contains forms of unconstrained English text acquired from a whiteboard. The performance of the RNN system using both raw and preprocessed input data is compared to that of an HMM based system using preprocessed data only [13]. To the best of our knowledge, this is the first time whole sentences of unconstrained handwriting have been directly transcribed from raw online data.

Section 2 describes the network architecture, the output layer and the algorithm for applying grammatical constraints. Section 3 provides experimental results, and conclusions are given in Section 4.

## 2 Method

### 2.1 Bidirectional Long Short-Term Memory

One of the key benefits of RNNs is their ability to make use of previous context. However, for standard RNN architectures, the range of context that can in practice be accessed is limited. The problem is that the influence of a given input on the hidden layer, and therefore on the network output, either decays or blows up exponentially as it cycles around the recurrent connections. This is often referred to as the *vanishing gradient problem* [4].

Long Short-Term Memory (LSTM; [5]) is an RNN architecture designed to address the vanishing gradient problem. An LSTM layer consists of multiple recurrently connected subnets, known as memory blocks. Each block contains a set of internal units, known as cells, whose activation is controlled by three multiplicative 'gate' units. The effect of the gates is to allow the cells to store and access information over long periods of time.

For many tasks it is useful to have access to future as well past context. Bidirectional RNNs [14] achieve this by presenting the input data forwards and backwards to two separate hidden layers, both of which are connected to the same output layer. Bidirectional LSTM (BLSTM) [3] combines the above architectures to provide access to long-range, bidirectional context.

### 2.2 Connectionist Temporal Classification

Connectionist temporal classification (CTC) [2] is an objective function designed for sequence labelling with RNNs. Unlike previous objective functions it does not require pre-segmented training data, or postprocessing to transform the network outputs into labellings. Instead, it trains the network to map directly from input sequences to the conditional probabilities of the possible labellings.

A CTC output layer contains one more unit than there are elements in the alphabet $L$ of labels for the task. The output activations are normalised with the softmax activation function [1]. At each time step, the first $|L|$ outputs are used to estimate the probabilities of observing the corresponding labels. The extra output estimates the probability of observing a 'blank', or no label. The combined output sequence estimates the joint probability of all possible alignments of the input sequence with all possible labellings. The probability of a particular labelling can then be estimated by summing over the probabilities of all the alignments that correspond to it.

More precisely, for an input sequence $\mathbf{x}$ of length $T$, choosing a label (or blank) at every time step according to the probabilities implied by the network outputs defines a probability distribution

over the set of length T sequences of labels and blanks. We denote this set $L'^T$, where $L' = L \cup \{blank\}$. To distinguish them from labellings, we refer to the elements of $L'^T$ as *paths*. Assuming that the label probabilities at each time step are conditionally independent given $\mathbf{x}$, the conditional probability of a path $\pi \in L'^T$ is given by

$$p(\pi|\mathbf{x}) = \prod_{t=1}^{T} y_{\pi_t}^t, \tag{1}$$

where $y_k^t$ is the activation of output unit $k$ at time $t$. Denote the set of sequences of length less than or equal to $T$ on the alphabet $L$ as $\mathbf{L}^{\leq T}$. Then Paths are mapped onto labellings $\mathbf{l} \in \mathbf{L}^{\leq T}$ by an operator $\mathcal{B}$ that removes first the repeated labels, then the blanks. For example, both $\mathcal{B}(a, -, a, b, -)$ and $\mathcal{B}(-, a, a, -, -, a, b, b)$ yield the labelling (a,a,b). Since the paths are mutually exclusive, the conditional probability of a given labelling $\mathbf{l} \in \mathbf{L}^{\leq T}$ is the sum of the probabilities of all paths corresponding to it:

$$p(\mathbf{l}|\mathbf{x}) = \sum_{\pi \in \mathcal{B}^{-1}(\mathbf{l})} p(\pi|\mathbf{x}). \tag{2}$$

Although a naive calculation of the above sum would be unfeasible, it can be efficiently evaluated with a graph-based algorithm [2], similar to the forward-backward algorithm for HMMs.

To allow for blanks in the output paths, for each label sequence $\mathbf{l} \in \mathbf{L}^{\leq T}$ consider a modified label sequence $\mathbf{l}' \in \mathbf{L}'^{\leq T}$, with blanks added to the beginning and the end and inserted between every pair of labels. The length of $\mathbf{l}'$ is therefore $|\mathbf{l}'| = 2|\mathbf{l}| + 1$.

For a labelling $\mathbf{l}$, define the *forward variable* $\alpha_t(s)$ as the summed probability of all paths whose length $t$ prefixes are mapped by $\mathcal{B}$ onto the length $s/2$ prefix of $\mathbf{l}$, i.e.

$$\alpha_t(s) = P(\pi_{1:t} : \mathcal{B}(\pi_{1:t}) = \mathbf{l}_{1:s/2}, \pi_t = \mathbf{l}'_s|\mathbf{x}) = \sum_{\substack{\pi: \\ \mathcal{B}(\pi_{1:t}) = \mathbf{l}_{1:s/2}}} \prod_{t'=1}^{t} y_{\pi_{t'}}^{t'}, \tag{3}$$

where, for some sequence $\mathbf{s}$, $\mathbf{s}_{a:b}$ is the subsequence $(\mathbf{s}_a, \mathbf{s}_{a+1}, ..., \mathbf{s}_{b-1}, \mathbf{s}_b)$, and $s/2$ is rounded down to an integer value.

The *backward variables* $\beta_t(s)$ are defined as the summed probability of all paths whose suffixes starting at $t$ map onto the suffix of $\mathbf{l}$ starting at label $s/2$

$$\beta_t(s) = P(\pi_{t+1:T} : \mathcal{B}(\pi_{t:T}) = \mathbf{l}_{s/2:|\mathbf{l}|}, \pi_t = \mathbf{l}'_s|\mathbf{x}) = \sum_{\substack{\pi: \\ \mathcal{B}(\pi_{t:T}) = \mathbf{l}_{s/2:|\mathbf{l}|}}} \prod_{t'=t+1}^{T} y_{\pi_{t'}}^{t'} \tag{4}$$

Both the forward and backward variables are calculated recursively [2]. The label sequence probability is given by the sum of the products of the forward and backward variables at any time step:

$$p(\mathbf{l}|\mathbf{x}) = \sum_{s=1}^{|\mathbf{l}'|} \alpha_t(s)\beta_t(s). \tag{5}$$

The objective function for CTC is the negative log probability of the network correctly labelling the entire training set. Let $S$ be a training set, consisting of pairs of input and target sequences $(\mathbf{x}, \mathbf{z})$, where target sequence $\mathbf{z}$ is at most as long as input sequence $\mathbf{x}$. Then the objective function is:

$$O^{CTC} = - \sum_{(\mathbf{x},\mathbf{z}) \in S} ln\left(p(\mathbf{z}|\mathbf{x})\right). \tag{6}$$

The network can be trained with gradient descent by differentiating $O^{CTC}$ with respect to the outputs, then using backpropagation through time to differentiate with respect to the network weights. Noting that the same label (or blank) may be repeated several times for a single labelling $l$, we define the set of positions where label $k$ occurs as $lab(\mathbf{l}, k) = \{s : \mathbf{l}'_s = k\}$, which may be empty. We then set $\mathbf{l} = \mathbf{z}$ and differentiate (5) with respect to the unnormalised network outputs $a_k^t$ to obtain:

$$\frac{\partial O^{CTC}}{\partial a_k^t} = -\frac{\partial ln\left(p(\mathbf{z}|\mathbf{x})\right)}{\partial a_k^t} = y_k^t - \frac{1}{p(\mathbf{z}|\mathbf{x})} \sum_{s \in lab(\mathbf{z},k)} \alpha_t(s)\beta_t(s). \tag{7}$$

Once the network is trained, we would ideally label some unknown input sequence $\mathbf{x}$ by choosing the most probable labelling $\mathbf{l}^*$:

$$\mathbf{l}^* = \arg \max_{\mathbf{l}} p(\mathbf{l}|\mathbf{x}). \tag{8}$$

Using the terminology of HMMs, we refer to the task of finding this labelling as *decoding*. Unfortunately, we do not know of a tractable decoding algorithm that is guaranteed to give optimal results. However a simple and effective approximation is given by assuming that the most probable path corresponds to the most probable labelling, i.e.

$$\mathbf{l}^* \approx \mathcal{B}\left(\arg \max_{\pi} p(\pi|\mathbf{x})\right). \tag{9}$$

## 2.3   Integration with an External Grammar

For some tasks we want to constrain the output labellings according to a predefined grammar. For example, in speech and handwriting recognition, the final transcriptions are usually required to form sequences of dictionary words. In addition it is common practice to use a language model to weight the probabilities of particular sequences of words.

We can express these constraints by altering the probabilities in (8) to be conditioned on some probabilistic grammar $G$, as well as the input sequence $\mathbf{x}$:

$$\mathbf{l}^* = \arg \max_{\mathbf{l}} p(\mathbf{l}|\mathbf{x}, G). \tag{10}$$

Absolute requirements, for example that $\mathbf{l}$ contains only dictionary words, can be incorporated by setting the probability of all sequences that fail to meet them to 0.

At first sight, conditioning on $G$ seems to contradict a basic assumption of CTC: that the labels are conditionally independent given the input sequences (see Eqn. (1)). Since the network attempts to model the probability of the whole labelling at once, there is nothing to stop it from learning inter-label transitions direct from the data, which would then be skewed by the external grammar. However, CTC networks are typically only able to learn local relationships such as commonly occurring pairs or triples of labels. Therefore as long as $G$ focuses on long range label interactions (such as the probability of one word following another when the outputs are letters) it doesn't interfere with the dependencies modelled by CTC.

The basic rules of probability tell us that $p(\mathbf{l}|\mathbf{x}, G) = \frac{p(\mathbf{l}|\mathbf{x})p(\mathbf{l}|G)p(\mathbf{x})}{p(\mathbf{x}|G)p(\mathbf{l})}$, where we have used the fact that $\mathbf{x}$ is conditionally independent of $G$ given $\mathbf{l}$. If we assume $\mathbf{x}$ is independent of G, this reduces to $p(\mathbf{l}|\mathbf{x}, G) = \frac{p(\mathbf{l}|\mathbf{x})p(\mathbf{l}|G)}{p(\mathbf{l})}$. That assumption is in general false, since both the input sequences and the grammar depend on the underlying generator of the data, for example the language being spoken. However it is a reasonable first approximation, and is particularly justifiable in cases where the grammar is created using data other than that from which $\mathbf{x}$ was drawn (as is common practice in speech and handwriting recognition, where independent textual corpora are used to generate language models).

Finally, if we assume that all label sequences are equally probable prior to any knowledge about the input or the grammar, we can drop the $p(\mathbf{l})$ term in the denominator to get

$$\mathbf{l}^* = \arg \max_{\mathbf{l}} p(\mathbf{l}|\mathbf{x})p(\mathbf{l}|G). \tag{11}$$

Note that, since the number of possible label sequences is finite (because both $L$ and $|\mathbf{l}|$ are finite), assigning equal prior probabilities does not lead to an improper prior.

We now describe an algorithm, based on the *token passing algorithm* for HMMs [16], that allows us to find an approximate solution to (11) for a simple grammar.

Let $G$ consist of a dictionary $D$ containing $W$ words, and a set of $W^2$ bigrams $p(w|\hat{w})$ that define the probability of making a transition from word $\hat{w}$ to word $w$. The probability of any labelling that does not form a sequence of dictionary words is 0.

For each word $w$, define the modified word $w'$ as $w$ with blanks added at the beginning and end and between each pair of labels. Therefore $|w'| = 2|w| + 1$. Define a token $tok = (score, history)$ to be a pair consisting of a real valued *score* and a *history* of previously visited words. In fact,

each token corresponds to a particular path through the network outputs, and its score is the log probability of that path. The basic idea of the token passing algorithm is to pass along the highest scoring tokens at every word state, then maximise over these to find the highest scoring tokens at the next state. The transition probabilities are used when a token is passed from the last state in one word to the first state in another. The output word sequence is given by the history of the highest scoring end-of-word token at the final time step.

At every time step $t$ of the length $T$ output sequence, each segment $s$ of each modified word $w'$ holds a single token $tok(w, s, t)$. This is the highest scoring token reaching that segment at that time. In addition we define the *input token* $tok(w, 0, t)$ to be the highest scoring token arriving at word $w$ at time $t$, and the *output token* $tok(w, -1, t)$ to be the highest scoring token leaving word $w$ at time $t$.

---

1: **Initialisation:**
2: **for** all words $w \in D$ **do**
3:     $tok(w, 1, 1) = (ln(y_b^1), (w))$
4:     $tok(w, 2, 1) = (ln(y_{w_1}^1), (w))$
5:     **if** $|w| = 1$ **then**
6:         $tok(w, -1, 1) = tok(w, 2, 1)$
7:     **else**
8:         $tok(w, -1, 1) = (-\infty, ())$
9:     $tok(w, s, 1) = (-\infty, ())$ for all $s \neq -1$
10: **Algorithm:**
11: **for** $t = 2$ to $T$ **do**
12:     sort output tokens $tok(w, -1, t - 1)$ by ascending score
13:     **for** all words $w \in D$ **do**
14:         $w^* = \arg \max_{\hat{w} \in D} tok(\hat{w}, -1, t - 1).score + ln\,(p(w|\hat{w}))$
15:         $tok(w, 0, t).score = tok(w^*, -1, t - 1).score + ln\,(p(w|w^*))$
16:         $tok(w, 0, t).history = tok(w^*, -1, t - 1).history + w$
17:         **for** segment $s = 1$ to $|w'|$ **do**
18:             $P = \{tok(w, s, t - 1), tok(w, s - 1, t - 1)\}$
19:             **if** $w'_s \neq blank$ and $s > 2$ and $w'_{s-2} \neq w'_s$ **then**
20:                 add $tok(w, s - 2, t - 1)$ to $P$
21:             $tok(w, s, t) = $ token in $P$ with highest score
22:             $tok(w, s, t).score \mathrel{+}= ln(y_{w'_s}^t)$
23:         $tok(w, -1, t) = $ highest scoring of $\{tok(w, |w'|, t), tok(w, |w'| - 1, t)\}$
24: **Termination:**
25: find output token $tok^*(w, -1, T)$ with highest score at time $T$
26: output $tok^*(w, -1, T).history$

**Algorithm 1:** CTC Token Passing Algorithm

---

The algorithm's worst case complexity is $O(TW^2)$, since line 14 requires a potential search through all $W$ words. However, because the output tokens $tok(w, -1, T)$ are sorted in order of score, the search can be terminated when a token is reached whose score is less than the current best score with the transition included. The typical complexity is therefore considerably lower, with a lower bound of $O(TWlogW)$ to account for the sort. If no bigrams are used, lines 14-16 can be replaced by a simple search for the highest scoring output token, and the complexity reduces to $O(TW)$.

Note that this is the same as the complexity of HMM decoding, if the search through bigrams is exhaustive. Much work has gone into developing more efficient decoding techniques (see e.g. [9]), typically by pruning improbable branches from the tree of labellings. Such methods are essential for applications where a rapid response is required, such as real time transcription. In addition, many decoders use more sophisticated language models than simple bigrams. Any HMM decoding algorithm could be applied to CTC outputs in the same way as token passing. However, we have stuck with a relatively basic algorithm since our focus here is on recognition rather than decoding.

# 3 Experiments

The experimental task was online handwriting recognition, using the IAM-OnDB handwriting database [12], which is available for public download from http://www.iam.unibe.ch/ fki/iamondb/

For CTC, we record both the character error rate, and the word error rate using Algorithm 1 with a language model and a dictionary. For the HMM system, the word error rate is quoted from the literature [13]. Both the character and word error rate are defined as the total number of insertions, deletions and substitutions in the algorithm's transcription of test set, divided by the combined length of the target transcriptions in the test set.

We compare results using both raw inputs direct from the pen sensor, and a preprocessed input representation designed for HMMs.

## 3.1 Data and Preprocessing

IAM-OnDB consists of pen trajectories collected from 221 different writers using a 'smart white-board' [12]. The writers were asked to write forms from the LOB text corpus [8], and the position of their pen was tracked using an infra-red device in the corner of the board. The input data consisted of the $x$ and $y$ pen coordinates, the points in the sequence when individual strokes (i.e. periods when the pen is pressed against the board) end, and the times when successive position measurements were made. Recording errors in the $x, y$ data were corrected by interpolating to fill in for missing readings, and removing steps whose length exceeded a certain threshold.

IAM-OnDB is divided into a training set, two validation sets, and a test set, containing respectively 5364, 1438, 1518 and 3859 written lines taken from 775, 192, 216 and 544 forms. The data sets contained a total of 3,298,424, 885,964, 1,036,803 and 2,425,5242 pen coordinates respectively. For our experiments, each line was used as a separate sequence (meaning that possible dependencies between successive lines were ignored).

The character level transcriptions contain 80 distinct target labels (capital letters, lower case letters, numbers, and punctuation). A dictionary consisting of the $20,000$ most frequently occurring words in the LOB corpus was used for decoding, along with a bigram language model optimised on the training and validation sets [13]. 5.6% of the words in the test set were not in the dictionary.

Two input representations were used. The first contained only the offset of the $x, y$ coordinates from the top left of the line, the time from the beginning of the line, and the marker for the ends of strokes. We refer to this as the *raw* input representation. The second representation used state-of-the-art preprocessing and feature extraction techniques [13]. We refer to this as the *preprocessed* input representation. Briefly, in order to account for the variance in writing styles, the pen trajectories were normalised with respect to such properties as the slant, skew and width of the letters, and the slope of the line as a whole. Two sets of input features were then extracted, the first consisting of 'online' features, such as pen position, pen speed, line curvature etc., and the second consisting of 'offline' features created from a two dimensional window of the image created by the pen.

## 3.2 Experimental Setup

The CTC network used the BLSTM architecture, as described in Section 2.1. The forward and backward hidden layers each contained 100 single cell memory blocks. The input layer was fully connected to the hidden layers, which were fully connected to themselves and the output layer. The output layer contained 81 units (80 characters plus the blank label). For the raw input representation, there were 4 input units and a total of 100,881 weights. For the preprocessed representation, there were 25 inputs and 117,681 weights. $tanh$ was used for the cell activation functions and logistic sigmoid in the range $[0, 1]$ was used for the gates. For both input representations, the data was normalised so that each input had mean 0 and standard deviation 1 on the training set. The network was trained with online gradient descent, using a learning rate of $10^{-4}$ and a momentum of 0.9. Training was stopped after no improvement was recorded on the validation set for 50 training epochs.

The HMM setup [13] contained a separate, left-to-right HMM with 8 states for each character ($8 * 81 = 648$ states in total). Diagonal mixtures of 32 Gaussians were used to estimate the observation

Table 1: **Word Error Rate (WER) on IAM-OnDB**. LM = language model. CTC results are a mean over 4 runs, $\pm$ standard error. All differences were significant ($p < 0.01$)

| System | Input | LM | WER |
|--------|-------|-----|------|
| HMM | preprocessed | ✓ | 35.5% [13] |
| CTC | raw | ✗ | $30.1 \pm 0.5\%$ |
| CTC | preprocessed | ✗ | $26.0 \pm 0.3\%$ |
| CTC | raw | ✓ | $22.8 \pm 0.2\%$ |
| CTC | preprocessed | ✓ | $20.4 \pm 0.3\%$ |

probabilities. All parameters, including the word insertion penalty and the grammar scale factor, were optimised on the validation set.

### 3.3 Results

The character error rate for the CTC network with the preprocessed inputs was $11.5 \pm 0.05\%$. From Table 1 we can see that with a dictionary and a language model this translates into a mean word error rate of 20.4%, which is a relative error reduction of 42.5% compared to the HMM. Without the language model, the error reduction was 26.8%. With the raw input data CTC achieved a character error rate of $13.9 \pm 0.1\%$, and word error rates that were close to those recorded with the preprocessed data, particularly when the language model was present.

The key difference between the input representations is that the raw data is less localised, and therefore requires more use of context. A useful indication of the network's sensitivity to context is provided by the derivatives of the output $y_k^t$ at a particular point $t$ in the data sequence with respect to the inputs $x_k^{t'}$ at all points $1 \le t' \le T$. We refer to these derivatives as the *sequential Jacobian*. Looking at the relative magnitude of the sequential Jacobian over time gives an idea of the range of context used, as illustrated in Figure 1.

## 4 Conclusion

We have combined a BLSTM CTC network with a probabilistic language model. We have applied this system to an online handwriting database and obtained results that substantially improve on a state-of-the-art HMM based system. We have also shown that the network's performance with raw sensor inputs is comparable to that with sophisticated preprocessing. As far as we are aware, our system is the first to successfully recognise unconstrained online handwriting using raw inputs only.

**Acknowledgments**

This research was funded by EC Sixth Framework project "NanoBioTact", SNF grant 200021-111968/1, and the SNF program "Interactive Multimodal Information Management (IM)2".

## References

[1] J. S. Bridle. Probabilistic interpretation of feedforward classification network outputs, with relationships to statistical pattern recognition. In F. Fogleman-Soulie and J.Herault, editors, *Neurocomputing: Algorithms, Architectures and Applications*, pages 227–236. Springer-Verlag, 1990.

[2] A. Graves, S. Fernández, F. Gomez, and J. Schmidhuber. Connectionist temporal classification: Labelling unsegmented sequence data with recurrent neural networks. In *Proc. 23rd Int. Conf. on Machine Learning*, Pittsburgh, USA, 2006.

[3] A. Graves and J. Schmidhuber. Framewise phoneme classification with bidirectional LSTM and other neural network architectures. *Neural Networks*, 18(5-6):602–610, June/July 2005.

[4] S. Hochreiter, Y. Bengio, P. Frasconi, and J. Schmidhuber. Gradient flow in recurrent nets: the difficulty of learning long-term dependencies. In S. C. Kremer and J. F. Kolen, editors, *A Field Guide to Dynamical Recurrent Neural Networks*. IEEE Press, 2001.

[5] S. Hochreiter and J. Schmidhuber. Long Short-Term Memory. *Neural Comp.*, 9(8):1735–1780, 1997.

[6] J. Hu, S. G. Lim, and M. K. Brown. Writer independent on-line handwriting recognition using an HMM approach. *Pattern Recognition*, 33:133–147, 2000.

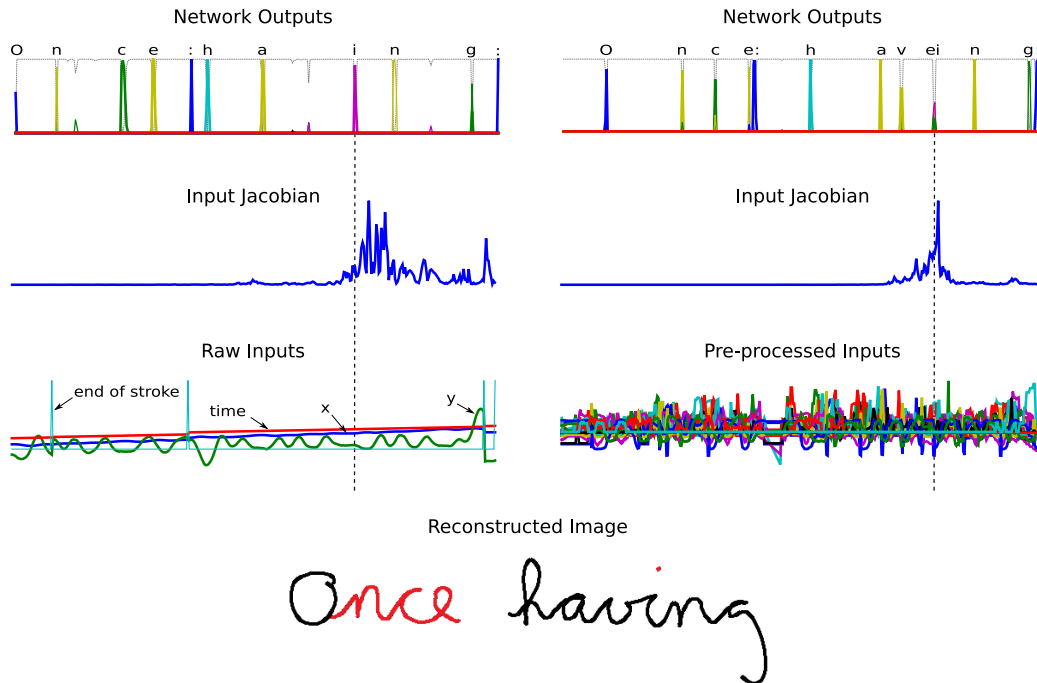

Figure 1: Sequential Jacobian for an excerpt from the IAM-OnDB, with raw inputs (left) and pre-processed inputs (right). For ease of visualisation, only the derivative with highest absolute value is plotted at each time step. The reconstructed image was created by plotting the pen coordinates recorded by the sensor. The individual strokes are alternately coloured red and black. For both representations, the Jacobian is plotted for the output corresponding to the label 'i' at the point when 'i' is emitted (indicated by the vertical dashed lines). Because bidirectional networks were used, the range of sensitivity extends in both directions from the dashed line. For the preprocessed data, the Jacobian is sharply peaked around the time when the output is emitted. For the raw data it is more spread out, suggesting that the network makes more use of long-range context. Note the spike in sensitivity to the very end of the raw input sequence: this corresponds to the delayed dot of the 'i'.

[7] S. Jaeger, S. Manke, J. Reichert, and A. Waibel. On-line handwriting recognition: the NPen++ recognizer. *Int. Journal on Document Analysis and Recognition*, 3:169–180, 2001.

[8] S. Johansson, R. Atwell, R. Garside, and G. Leech. The tagged LOB corpus user's manual; Norwegian Computing Centre for the Humanities, 1986.

[9] P. Lamere, P. Kwok, W. Walker, E. Gouvea, R. Singh, B. Raj, and P. Wolf. Design of the CMU Sphinx-4 decoder. In *Proc. 8th European Conf. on Speech Communication and Technology*, Aug. 2003.

[10] Y. LeCun, L. Bottou, Y. Bengio, and P. Haffner. Gradient-based learning applied to document recognition. *Proc. IEEE*, 86(11):2278–2324, Nov. 1998.

[11] Y. LeCun, F. Huang, and L. Bottou. Learning methods for generic object recognition with invariance to pose and lighting. In *Proc. of CVPR'04*. IEEE Press, 2004.

[12] M. Liwicki and H. Bunke. IAM-OnDB - an on-line English sentence database acquired from handwritten text on a whiteboard. In *Proc. 8th Int. Conf. on Document Analysis and Recognition*, volume 2, pages 956–961, 2005.

[13] M. Liwicki, A. Graves, S. Fernández, H. Bunke, and J. Schmidhuber. A novel approach to on-line handwriting recognition based on bidirectional long short-term memory networks. In *Proc. 9th Int. Conf. on Document Analysis and Recognition*, Curitiba, Brazil, Sep. 2007.

[14] M. Schuster and K. K. Paliwal. Bidirectional recurrent neural networks. *IEEE Transactions on Signal Processing*, 45:2673–2681, Nov. 1997.

[15] P. Y. Simard, D. Steinkraus, and J. C. Platt. Best practices for convolutional neural networks applied to visual document analysis. In *Proc. 7th Int. Conf. on Document Analysis and Recognition*, page 958, Washington, DC, USA, 2003. IEEE Computer Society.

[16] S. Young, N. Russell, and J. Thornton. Token passing: A simple conceptual model for connected speech recognition systems. Technical Report CUED/F-INFENG/TR38, Cambridge University Eng. Dept., 1989.

